# Dual Averaging Method for Regularized Stochastic Learning and Online Optimization

**Lin Xiao**
Microsoft Research, Redmond, WA 98052
`lin.xiao@microsoft.com`

## Abstract

We consider regularized stochastic learning and online optimization problems, where the objective function is the sum of two convex terms: one is the loss function of the learning task, and the other is a simple regularization term such as $\ell_1$-norm for promoting sparsity. We develop a new online algorithm, the *regularized dual averaging* (RDA) method, that can explicitly exploit the regularization structure in an online setting. In particular, at each iteration, the learning variables are adjusted by solving a simple optimization problem that involves the running average of all past subgradients of the loss functions and the whole regularization term, not just its subgradient. Computational experiments show that the RDA method can be very effective for sparse online learning with $\ell_1$-regularization.

## 1 Introduction

In machine learning, online algorithms operate by repetitively drawing random examples, one at a time, and adjusting the learning variables using simple calculations that are usually based on the single example only. The low computational complexity (per iteration) of online algorithms is often associated with their slow convergence and low accuracy in solving the underlying optimization problems. As argued in [1, 2], the combined low complexity and low accuracy, together with other tradeoffs in statistical learning theory, still make online algorithms a favorite choice for solving large-scale learning problems. Nevertheless, traditional online algorithms, such as stochastic gradient descent (SGD), has limited capability of exploiting problem structure in solving *regularized* learning problems. As a result, their low accuracy often makes it hard to obtain the desired regularization effects, e.g., sparsity under $\ell_1$-regularization. In this paper, we develop a new online algorithm, the *regularized dual averaging* (RDA) method, that can explicitly exploit the regularization structure in an online setting. We first describe the two types of problems addressed by the RDA method.

### 1.1 Regularized stochastic learning

The regularized stochastic learning problems we consider are of the following form:

$$\underset{w}{\text{minimize}} \quad \left\{ \phi(w) \triangleq \mathbf{E}_z f(w, z) + \Psi(w) \right\} \tag{1}$$

where $w \in \mathbf{R}^n$ is the optimization variable (called *weights* in many learning problems), $z = (x, y)$ is an input-output pair drawn from an (unknown) underlying distribution, $f(w, z)$ is the loss function of using $w$ and $x$ to predict $y$, and $\Psi(w)$ is a regularization term. We assume $f(w, z)$ is convex in $w$ for each $z$, and $\Psi(w)$ is a closed convex function. Examples of the loss function $f(w, z)$ include:

- Least-squares: $x \in \mathbf{R}^n$, $y \in \mathbf{R}$, and $f(w, (x,y)) = (y - w^T x)^2$.
- Hinge loss: $x \in \mathbf{R}^n$, $y \in \{+1, -1\}$, and $f(w, (x,y)) = \max\{0, 1 - y(w^T x)\}$.
- Logistic regression: $x \in \mathbf{R}^n$, $y \in \{+1, -1\}$, and $f(w, (x,y)) = \log\left(1 + \exp\left(-y(w^T x)\right)\right)$.

Examples of the regularization term $\Psi(w)$ include:

- $\ell_1$-regularization: $\Psi(w) = \lambda\|w\|_1$ with $\lambda > 0$. With $\ell_1$-regularization, we hope to get a relatively sparse solution, i.e., with many entries of $w$ being zeroes.

- $\ell_2$-regularization: $\Psi(w) = (\sigma/2)\|w\|_2^2$, for some $\sigma > 0$.

- Convex constraints: $\Psi(w)$ is the *indicator function* of a closed convex set $C$, i.e., $\Psi(w) = 0$ if $w \in C$ and $+\infty$ otherwise.

In this paper, we focus on *online algorithms* that process samples sequentially as they become available. Suppose at time $t$, we have the most up-to-date weight $w_t$. Whenever $z_t$ is available, we can evaluate the loss $f(w_t, z_t)$, and a subgradient $g_t \in \partial f(w_t, z_t)$ (here $\partial f(w, z)$ denotes the subdifferential of $f$ with respect to $w$). Then we compute the new weight $w_{t+1}$ based on these information. For solving the problem (1), the standard *stochastic gradient descent* (SGD) method takes the form

$$w_{t+1} = w_t - \alpha_t \left(g_t + \xi_t\right), \tag{2}$$

where $\alpha_t$ is an appropriate stepsize, and $\xi_t$ is a subgradient of $\Psi$ at $w_t$. The SGD method has been very popular in the machine learning community due to its capability of scaling with large data sets and good generalization performance observed in practice (e.g., [3, 4]).

Nevertheless, a main drawback of the SGD method is its lack of capability in exploiting problem structure, especially for *regularized* learning problems. As a result, their low accuracy (compared with interior-point method for batch optimization) often makes it hard to obtain the desired regularization effect. An important example and motivation for this paper is $\ell_1$-regularized stochastic learning, where $\Psi(w) = \lambda\|w\|_1$. Even with relatively big $\lambda$, the SGD method (2) usually does not generate sparse solutions because only in very rare cases two float numbers add up to zero. Various methods for rounding or truncating the solutions are proposed to generate sparse solutions (e.g., [5]).

Inspired by recently developed first-order methods for optimizing composite functions [6, 7, 8], the *regularized dual averaging* (RDA) method we develop exploits the full regularization structure at each online iteration. In other words, at each iteration, the learning variables are adjusted by solving a simple optimization problem that involves the whole regularization term, not just its subgradients. For many practical learning problems, we actually are able to find a closed-form solution for the auxiliary optimization problem at each iteration. This means that the computational complexity per iteration is $O(n)$, the same as the SGD method. Moreover, the RDA method converges to the optimal solution of (1) with the optimal rate $O(1/\sqrt{t})$. If the the regularization function $\Psi(w)$ is strongly convex, we have the better rate $O(\ln t/t)$ by setting appropriate parameters in the algorithm.

## 1.2 Regularized online optimization

In *online optimization* (e.g., [9]), we make a sequence of decision $w_t$, for $t = 1, 2, 3, \ldots$. At each time $t$, a previously unknown cost function $f_t$ is revealed, and we encounter a loss $f_t(w_t)$. We assume that the functions $f_t$ are convex for all $t \geq 1$. The goal of an online algorithm is to ensure that the total cost up to each time $t$, $\sum_{\tau=1}^{t} f_t(w_t)$, is not much larger than $\min_w \sum_{\tau=1}^{t} f_t(w)$, the smallest total cost of any fixed decision $w$ from hindsight. The difference between these two cost is called the *regret* of the online algorithm. Applications of online optimization include online prediction of time series and sequential investment (e.g. [10]).

In regularized online optimization, we add to each cost function a convex regularization function $\Psi(w)$. For any fixed decision variable $w$, consider the *regret*

$$R_t(w) \triangleq \sum_{\tau=1}^{t} \left(f_\tau(w_\tau) + \Psi(w_\tau)\right) - \sum_{\tau=1}^{t} \left(f_\tau(w) + \Psi(w)\right). \tag{3}$$

The RDA method we develop can also be used to solve the above regularized online optimization problem, and it has an $O(\sqrt{t})$ regret bound. Again, if the regularization term $\Psi(w)$ is strongly convex, the regret bound is $O(\ln t)$. However, the main advantage of the RDA method, compared with other online algorithms, is its explicit regularization effect at each iteration.

---

**Algorithm 1** Regularized dual averaging (RDA) method

---

**input:**

- a strongly convex function $h(w)$ with modulus 1 on $\mathrm{dom}\Psi$, and $w_0 \in \mathbf{R}^n$, such that

$$w_0 = \arg\min_w h(w) \in \mathrm{Arg}\min_w \Psi(w). \tag{4}$$

- a pre-determined nonnegative and nondecreasing sequence $\beta_t$ for $t \geq 1$.

**initialize:** $w_1 = w_0, \bar{g}_0 = 0$.

**for** $t = 1, 2, 3, \ldots$ **do**

  1. Given the function $f_t$, compute a subgradient $g_t \in \partial f_t(w_t)$.
  2. Update the average subgradient $\bar{g}_t$:

$$\bar{g}_t = \frac{t-1}{t}\bar{g}_{t-1} + \frac{1}{t}g_t \tag{5}$$

  3. Compute the next iterate $w_{t+1}$:

$$w_{t+1} = \arg\min_w \left\{ \langle \bar{g}_t, w \rangle + \Psi(w) + \frac{\beta_t}{t}h(w) \right\} \tag{6}$$

**end for**

---

## 2 Regularized dual averaging method

In this section, we present the generic RDA method (Algorithm 1) for solving regularized stochastic learning and online optimization problems, and give some concrete examples. To unify notation, we write $f(w, z_t)$ as $f_t(w)$ for stochastic learning problems. The RDA method uses an auxiliary strongly convex function $h(w)$. A function $h$ is called *strongly convex* with respect to a norm $\|\cdot\|$ if there exists a constant $\sigma > 0$ such that

$$h(\alpha w + (1 - \alpha)u) \leq \alpha h(w) + (1 - \alpha)h(u) - \frac{\sigma}{2}\alpha(1 - \alpha)\|w - u\|^2, \tag{7}$$

for all $w, u \in \mathrm{dom}h$. The constant $\sigma$ is called the *convexity parameter*, or the *modulus* of strong convexity. In equation (4), $\mathrm{Arg}\min_w \Psi(w)$ denotes the convex set of minimizers of $\Psi$.

In Algorithm 1, step 1 is to compute a subgradient of $f_t$ at $w_t$, which is standard for all (sub)gradient-based methods. Step 2 is the online version of computing average gradient $\bar{g}_t$ (dual average). In step 3, we assume that the functions $\Psi$ and $h$ are *simple*, meaning that the minimization problem in (6) can be solved with litter effort, especially if we are able to find a closed-form solution for $w_{t+1}$. This assumption seems to be restrictive. But the following examples show that this indeed is the case for many important learning problems in practice.

If the regularization function $\Psi(w)$ has convexity parameter $\sigma = 0$ (i.e., it is not strongly convex), we can choose a parameter $\gamma > 0$ and use the sequence

$$\beta_t = \gamma\sqrt{t}, \qquad t = 1, 2, 3, \ldots \tag{8}$$

to obtain an $O(1/\sqrt{t})$ convergence rate for stochastic learning, or an $O(\sqrt{t})$ regret bound for online optimization. The formal convergence theorems are given in Sections 3. Here are some examples:

- *Nesterov's dual averaging method.* Let $\Psi(w)$ be the indicator function of a close convex set $C$. This recovers the method of [11]: $w_{t+1} = \arg\min_{w \in C}\left\{\langle \bar{g}_t, w \rangle + (\gamma/\sqrt{t})h(w)\right\}$.

- $\ell_1$-*regularization:* $\Psi(w) = \lambda\|w\|_1$ for some $\lambda > 0$. In this case, let $w_0 = 0$ and

$$h(w) = \frac{1}{2}\|w\|_2^2 + \rho\|w\|_1,$$

where $\rho \geq 0$ is a *sparsity enhancing* parameter. The solution to (6) can be found as

$$w_{t+1}^{(i)} = \begin{cases} 0 & \text{if } \left|\bar{g}_t^{(i)}\right| \leq \lambda_t^{\mathrm{RDA}}, \\ -\frac{\sqrt{t}}{\gamma}\left(\bar{g}_t^{(i)} - \lambda_t^{\mathrm{RDA}}\,\mathrm{sign}\big(\bar{g}_t^{(i)}\big)\right) & \text{otherwise}, \end{cases} \quad i = 1, \ldots, n, \tag{9}$$

where $\lambda_t^{\mathrm{RDA}} = \lambda + \rho/\sqrt{t}$. Notice that the truncating threshold $\lambda_t$ is at least as large as $\lambda$. This is the main difference of our method from related work, see Section 4.

If the regularization function $\Psi(w)$ has convexity parameter $\sigma > 0$, we can use any nonnegative, nondecreasing sequence $\{\beta_t\}_{t\geq 1}$ that is dominated by $\ln t$, to obtain an $O(\ln t/\sqrt{t})$ convergence rate for stochastic learning, or an $O(\ln t)$ regret bound for online optimization (see Section 3). For simplicity, in the following examples, we use $\beta_t = 0$ for all $t \geq 1$, and we do not need $h(w)$.

- *Mixed $\ell_1/\ell_2^2$-regularization.* Let $\Psi(w) = \lambda\|w\|_1 + (\sigma/2)\|w\|_2^2$ with $\lambda$, $\sigma > 0$. Then

$$
w_{t+1}^{(i)} = \begin{cases} 0 & \text{if } \left|\bar{g}_t^{(i)}\right| \leq \lambda, \\ -\dfrac{1}{\sigma}\left(\bar{g}_t^{(i)} - \lambda\operatorname{sign}\left(\bar{g}_t^{(i)}\right)\right) & \text{otherwise,} \end{cases} \qquad i = 1, \ldots, n.
$$

  Of course, setting $\lambda = 0$ gives the algorithm for pure $\ell_2^2$-regularization.

- *Kullback-Leibler (KL) divergence regularization:* $\Psi(w) = \sigma D_{\mathrm{KL}}(w\|p)$, where $w$ lies in the standard simplex, $p$ is a given probability distribution, and

$$
D_{\mathrm{KL}}(w\|p) \triangleq \sum_{i=1}^{n} w^{(i)} \ln\left(\frac{w^{(i)}}{p^{(i)}}\right).
$$

  Note that $D_{\mathrm{KL}}(w\|p)$ is strongly convex with respect to $\|w\|_1$ with modulus 1 (e.g., [12]). In this case,

$$
w_{t+1}^{(i)} = \frac{1}{Z_{t+1}} p^{(i)} \exp\left(-\frac{1}{\sigma}\bar{g}_t^{(i)}\right),
$$

  where $Z_{t+1}$ is a normalization parameter such that $\sum_{i=1}^{n} w_{t+1}^{(i)} = 1$.

## 3 Regret bounds and convergence rates

We first give bounds on the regret $R_t(w)$ defined in (3), when the RDA method is used for solving regularized online optimization problem. To simplify notations, we define the following sequence:

$$
\Delta_t \triangleq (\beta_0 - \beta_1)h(w_2) + \beta_t D^2 + \frac{L^2}{2}\sum_{\tau=0}^{t-1}\frac{1}{\sigma\tau + \beta_\tau}, \qquad t = 1, 2, 3, \ldots, \tag{10}
$$

where $D$ and $L$ are some given constants, $\sigma$ is the convexity parameter of the regularization function $\Psi(w)$, and $\{\beta_\tau\}_{\tau=1}^{t}$ is the input sequence to the RDA method, which is nonnegative and nondecreasing. Notice that we just introduced an extra parameter $\beta_0$. We require $\beta_0 > 0$ to avoid blowup of the first term (when $\tau = 0$) in the summation in (10). This parameter does not appear in Algorithm 1, instead, it is solely for the convenience of convergence analysis. In fact, whenever $\beta_1 > 0$, we can set $\beta_0 = \beta_1$, so that the term $(\beta_0 - \beta_1)h(w_2)$ vanishes. We also note that $w_2$ is determined at the end of the step $t = 1$, so $\Delta_1$ is well defined. Finally, for any given constant $D > 0$, we define

$$
\mathcal{F}_D \triangleq \left\{w \in \operatorname{dom}\Psi \mid h(w) \leq D^2\right\}.
$$

**Theorem 1** *Let the sequences $\{w_\tau\}_{\tau=1}^{t}$ and $\{g_\tau\}_{\tau=1}^{t}$ be generated by Algorithm 1. Assume there is a constant $L$ such that $\|g_t\|_* \leq L$ for all $t \geq 1$, where $\|\cdot\|_*$ is the dual norm of $\|\cdot\|$. Then for any $t \geq 1$ and any $w \in \mathcal{F}_D$, we have*

$$
R_t(w) \leq \Delta_t. \tag{11}
$$

The proof of this theorem is given in the longer version of this paper [13]. Here we give some direct consequences based on concrete choices of algorithmic parameters.

If the regularization function $\Psi(w)$ has convexity parameter $\sigma = 0$, then the sequence $\{\beta_t\}_{t\geq 1}$ defined in (8) together with $\beta_0 = \beta_1$ lead to

$$
\Delta_t = \gamma\sqrt{t}D^2 + \frac{L^2}{2\gamma}\left(1 + \sum_{\tau=1}^{t-1}\frac{1}{\sqrt{\tau}}\right) \leq \gamma\sqrt{t}D^2 + \frac{L^2}{2\gamma}\left(1 + \left(2\sqrt{t} - 2\right)\right) \leq \left(\gamma D^2 + \frac{L^2}{\gamma}\right)\sqrt{t}.
$$

The best $\gamma$ that minimizes the above bound is $\gamma^\star = L/D$, which leads to

$$
R_t(w) \leq 2LD\sqrt{t}. \tag{12}
$$

If the regularization function $\Psi(w)$ is strongly convex, i.e., with a convexity parameter $\sigma > 0$, then any nonnegative, nondecreasing sequence that is dominated by $\ln t$ will give an $O(\ln t)$ regret bound. We can simply choose $h(w) = (1/\sigma)\Psi(w)$ whenever needed. Here are several possibilities:

- *Positive constant sequences.* For simplicity, let $\beta_t = \sigma$ for $t \geq 1$ and $\beta_0 = \beta_1$. In this case,

$$\Delta_t = \sigma D^2 + \frac{L^2}{2\sigma} \sum_{\tau=1}^{t} \frac{1}{\tau} \leq \sigma D^2 + \frac{L^2}{2\sigma}(1 + \ln t).$$

- *The logrithmic sequence.* Let $\beta_t = \sigma(1 + \ln t)$ for $t \geq 1$, and $\beta_0 = \sigma$. In this case,

$$\Delta_t = \sigma(1 + \ln t)D^2 + \frac{L^2}{2\sigma}\left(1 + \sum_{\tau=1}^{t-1} \frac{1}{\tau + 1 + \ln \tau}\right) \leq \left(\sigma D^2 + \frac{L^2}{2\sigma}\right)(1 + \ln t).$$

- *The zero sequence* $\beta_t = 0$ *for* $t \geq 1$, *with* $\beta_0 = \sigma$. Using $h(w) = (1/\sigma)\Psi(w)$, we have

$$\Delta_t \leq \Psi(w_2) + \frac{L^2}{2\sigma}\left(1 + \sum_{\tau=1}^{t} \frac{1}{\tau}\right) \leq \frac{L^2}{2\sigma}(6 + \ln t),$$

where we used $\Psi(w_2) \leq 2L^2/\sigma$, as proved in [13]. This bound does not depend on $D$.

When Algorithm 1 is used to solve regularized stochastic learning problems, we have the following:

**Theorem 2** *Assume there exists an optimal solution $w^\star$ to the problem (1) that satisfies $h(w^\star) \leq D^2$ for some $D > 0$, and there is an $L > 0$ such that $\mathbf{E}\|g\|_*^2 \leq L^2$ for all $g \in \partial f(w, z)$ and $w \in \mathrm{dom}\Psi$. Then for any $t \geq 1$, we have*

$$\mathbf{E}\,\phi(\bar{w}_t) - \phi(w^\star) \leq \frac{\Delta_t}{t}, \qquad \textit{where} \qquad \bar{w}_t = \frac{1}{t} \sum_{\tau=1}^{t} w_\tau.$$

The proof of Theorem 2 is given in [13]. Further analysis for the cases $\sigma = 0$ and $\sigma > 0$ are the same as before. We only need to divide every regret bound by $t$ to obtain the convergence rate.

## 4  Related work

There have been several recent work that address online algorithms for regularized learning problems, especially with $\ell_1$-regularization; see, e.g., [14, 15, 16, 5, 17]. In particular, a forward-backward splitting method (FOBOS) is studied in [17] for solving the same problems we consider. In an online setting, each iteration of the FOBOS method can be written as

$$w_{t+1} = \arg\min_w \left\{ \frac{1}{2}\|w - (w_t - \alpha_t g_t)\|^2 + \alpha_t \Psi(w) \right\}, \tag{13}$$

where $\alpha_t$ is set to be $O(1/\sqrt{t})$ if $\Psi(w)$ has convexity parameter $\sigma = 0$, and $O(1/t)$ if $\sigma > 0$. The RDA method and FOBOS use very different weights on the regularization term $\Psi(w)$: RDA in (6) uses the original $\Psi(w)$ without any scaling, while FOBOS scales $\Psi(w)$ by a diminishing stepsize $\alpha_t$.

The difference is more clear in the special case of $\ell_1$-regularization, i.e., when $\Psi(w) = \lambda\|w\|_1$. For this purpose, we consider the *Truncated Gradient* (TG) method proposed in [5]. The TG method truncates the solutions obtained by the standard SGD method with an integer period $K \geq 1$. More specifically, each component of $w_t$ is updated as

$$w_{t+1}^{(i)} = \begin{cases} \mathrm{trnc}\left(w_t^{(i)} - \alpha_t g_t^{(i)}, \lambda_t^{\mathrm{TG}}, \theta\right) & \text{if } \mathrm{mod}(t, K) = 0, \\ w_t^{(i)} - \alpha_t g_t^{(i)} & \text{otherwise.} \end{cases} \tag{14}$$

where $\lambda_t^{\mathrm{TG}} = \alpha_t \lambda K$, the function $\mathrm{mod}(t, K)$ means the remainder on division of $t$ by $K$, and

$$\mathrm{trnc}(\omega, \lambda_t^{\mathrm{TG}}, \theta) = \begin{cases} 0 & \text{if } |\omega| \leq \lambda_t^{\mathrm{TG}}, \\ \omega - \lambda_t^{\mathrm{TG}} \,\mathrm{sign}(\omega) & \text{if } \lambda_t^{\mathrm{TG}} < |\omega| \leq \theta, \\ \omega & \text{if } |\omega| > \theta. \end{cases}$$

When $K = 1$ and $\theta = +\infty$, the TG method is the same as the FOBOS method (13). Now comparing the truncation thresholds $\lambda_t^{\mathrm{TG}}$ and $\lambda_t^{\mathrm{RDA}}$ used in (9): with $\alpha_t = O(1/\sqrt{t})$, we have $\lambda_t^{\mathrm{TG}} = O(1/\sqrt{t})\lambda_t^{\mathrm{RDA}}$. Therefore, the RDA method can generate much more sparse solutions. This is confirmed by our computational experiments in Section 5.

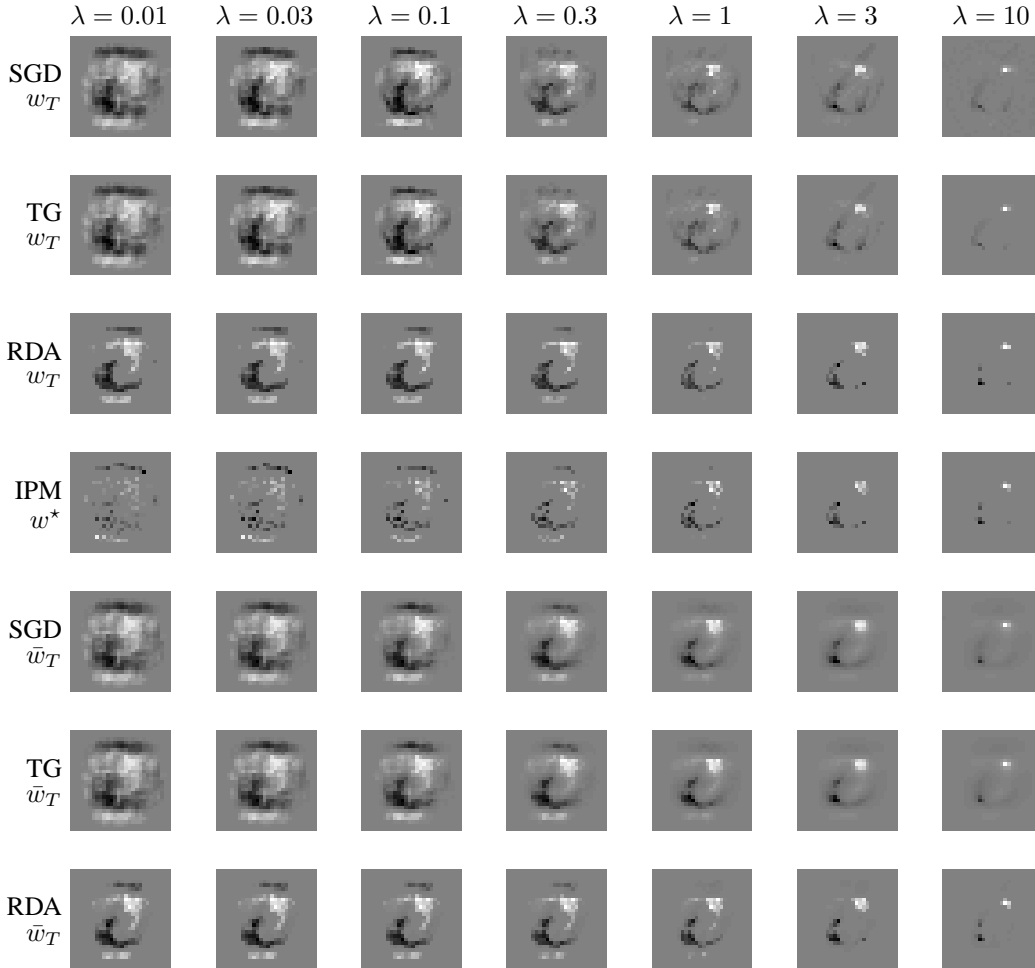

Figure 1: Sparsity patterns of the weight $w_T$ and the average weight $\bar{w}_T$ for classifying the digits 6 and 7 when varying the regularization parameter $\lambda$ from 0.01 to 10. The background gray represents the value zero, bright spots represent positive values and dark spots represent negative values.

## 5   Computational experiments

We provide computational experiments for the $\ell_1$-RDA method using the MNIST dataset of hand-written digits [18]. Each image from the dataset is represented by a $28 \times 28$ gray-scale pixel-map, for a total of 784 features. Each of the 10 digits has roughly 6,000 training examples and 1,000 testing examples. No preprocessing of the data is employed.

We use $\ell_1$-regularized logistic regression to do binary classification on each of the 45 pairs of digits. In the experiments, we compare the $\ell_1$-RDA method (9) with the SGD method (2) and the TG/FOBOS method (14) with $\theta = \infty$. These three online algorithms have similar convergence rate and the same order of computational complexity per iteration. We also compare them with the batch optimization approach, using an efficient interior-point method (IPM) developed by [19].

Each pair of digits have about 12,000 training examples and 2,000 testing examples. We use online algorithms to go through the (randomly permuted) data only once, therefore the algorithms stop at $T = 12,000$. We vary the regularization parameter $\lambda$ from 0.01 to 10. As a reference, the maximum $\lambda$ for the batch optimization case [19] is mostly in the range of $30 - 50$ (beyond which the optimal weights are all zeros). In the $\ell_1$-RDA method (9), we use $\gamma = 5,000$, and set $\rho = 0$ for basic regularization, or $\rho = 0.005$ (effectively $\gamma\rho = 25$) for enhanced regularization effect. The tradeoffs in choosing these parameters are further investigated in [13]. For the SGD and TG methods, we use a constant stepsize $\alpha = (1/\gamma)\sqrt{2/T}$. When $\gamma = L/D$, which gives the best convergence bound (12)

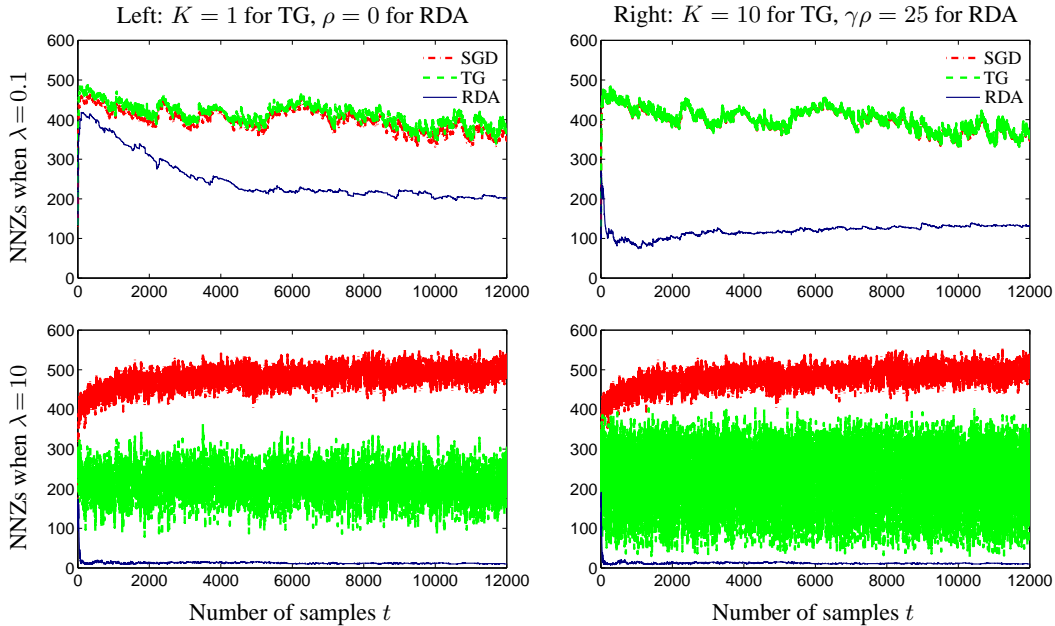

Figure 2: Number of non-zeros (NNZs) in $w(t)$ for the three online algorithms (classifying 6 and 7).

for the RDA method, the corresponding $\alpha = (D/L)\sqrt{2/T}$ also gives the best convergence rate for the SGD method (e.g., [20]). In the TG method, the truncation period is set to $K = 1$ for basic regularization, or $K = 10$ for enhanced regularization effect, as suggested in [5].

Figure 1 shows the sparsity patterns of the solutions $w_T$ and $\bar{w}_T$ for classifying the digits 6 and 7. Both the TG and RDA methods were run with parameters for enhanced $\ell_1$-regularization: $K = 10$ for TG and $\gamma\rho = 25$ for RDA. The sparsity patterns obtained by the RDA method are most close to the batch optimization results solved by IPM, especially for larger $\lambda$.

Figure 2 plots the number of non-zeros (NNZs) in $w(t)$ for different online algorithms. Only the RDA method and TG with $K = 1$ give explicit zero weights at every step. In order to count the NNZs in all other cases, we set a small threshold for rounding the weights to zero. Considering that the magnitudes of the largest weights in Figure 1 are mostly on the order of $10^{-3}$, we set $10^{-5}$ as the threshold and verified that rounding elements less than $10^{-5}$ to zero does not affect the testing errors. Note that we do not truncate the weights for RDA and TG with $K = 1$ further, even if some of their components are below $10^{-5}$. It can be seen that the RDA method maintains a much more sparse $w(t)$ than the other two online algorithms. While the TG method generate more sparse solutions than the SGD method when $\lambda$ is large, the NNZs in $w(t)$ oscillates with a very big range. In contrast, the RDA method demonstrate a much more smooth variation in the NNZs.

Figure 3 illustrates the tradeoffs between sparsity and testing error rates for classifying 6 and 7. Since the performance of the online algorithms vary when the training data are given in different permutations, we run them on 100 randomly permuted sequences of the same training set, and plot the means and standard deviations shown as error bars. For the SGD and TG methods, the testing error rates of $w_T$ vary a lot for different random sequences. In contrast, the RDA method demonstrates very robust performance (small standard deviations) for $w_T$, even though the theorems only give performance bound for the averaged weight $\bar{w}_T$. Note that $\bar{w}_T$ obtained by SGD and TG have much smaller error rates than those of RDA and batch optimization, especially for larger $\lambda$. The explanation is that these lower error rates are obtained with much more nonzero features.

Figure 4 shows summary of classification results for all the 45 pairs of digits. For clarity of presentation, here we only plot results of the $\ell_1$-RDA method and batch optimization using IPM. (The NNZs obtained by SGD and TG are mostly above the limit of the vertical axes, which is set at 200). We see that, overall, the solutions obtained by the $\ell_1$-RDA method demonstrate very similar tradeoffs between sparsity and testing error rates as rendered by the batch optimization solutions.

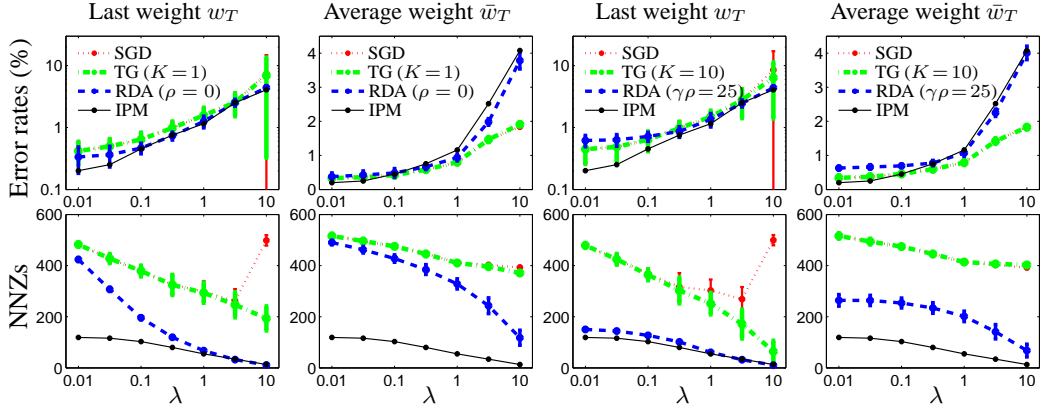

Figure 3: Tradeoffs between testing error rates and NNZs in solutions (for classifying 6 and 7).

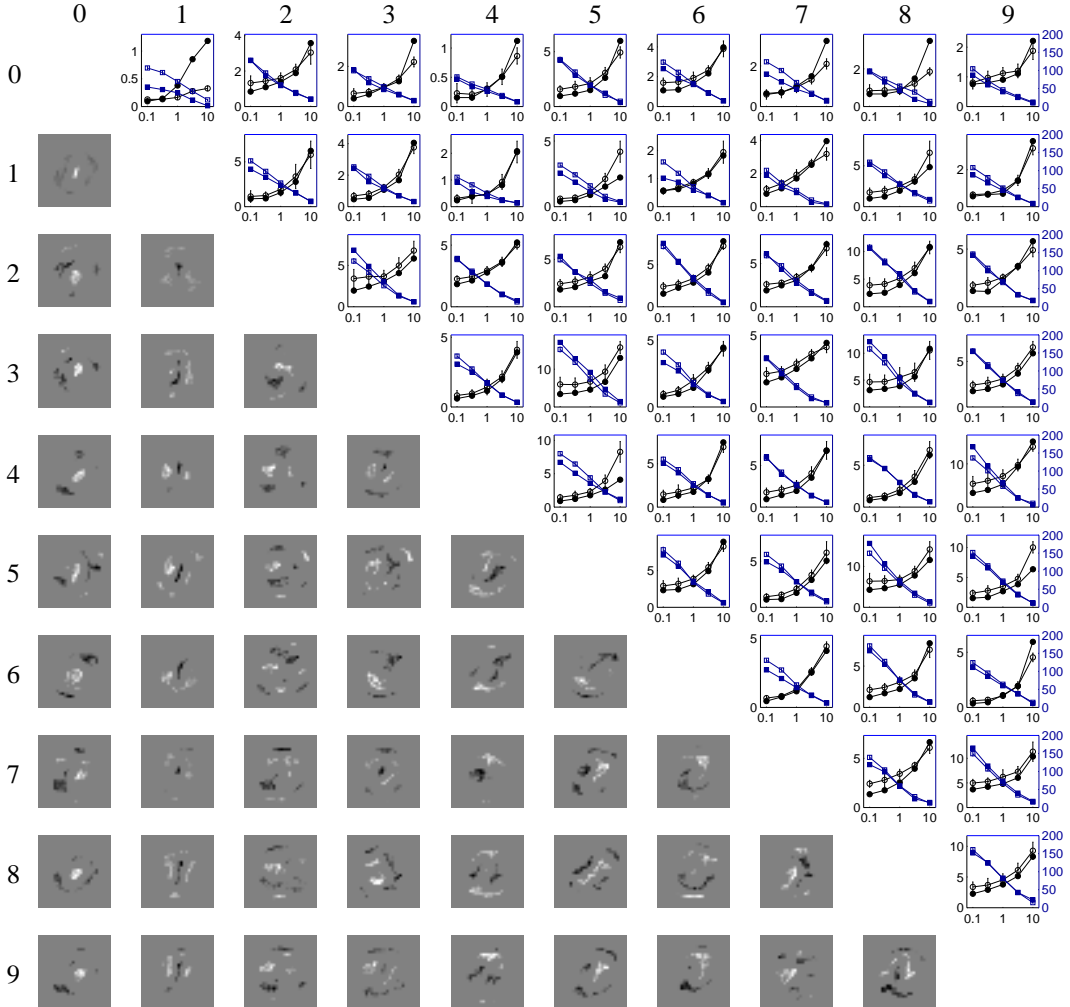

Figure 4: Binary classification for all 45 pairs of digits. The images in the lower-left triangular area show sparsity patterns of $w_T$ with $\lambda = 1$, obtained by the $\ell_1$-RDA with $\gamma\rho = 25$. The plots in the upper-right triangular area show tradeoffs between sparsity and testing error rates, by varying $\lambda$ from 0.1 to 10. The solid circles and solid squares show error rates and NNZs in $w_T$, respectively, using IPM for batch optimization. The hollow circles and hollow squares show error rates and NNZs of $w_T$, respectively, using the $\ell_1$-RDA method. The vertical bars centered at hollow circles and squares show standard deviations by running on 100 random permutations of the training data.

# References

[1] L. Bottou and O. Bousquet. The tradeoffs of large scale learning. In J.C. Platt, D. Koller, Y. Singer, and S. Roweis, editors, *Advances in Neural Information Processing Systems 20*, pages 161–168. MIT Press, Cambridge, MA, 2008.

[2] S. Shalev-Shwartz and N. Srebro. SVM optimization: Inverse dependence on training set size. In *Proceedings of the 25th International Conference on Machine Learning (ICML)*, 2008.

[3] L. Bottou and Y. LeCun. Large scale online learning. In S. Thrun, L. Saul, and B. Schölkopf, editors, *Advances in Neural Information Processing Systems 16*. MIT Press, Cambridge, MA, 2004.

[4] T. Zhang. Solving large scale linear prediction problems using stochastic gradient descent algorithms. In *Proceedings of the 21st International Conference on Machine Learning (ICML)*, Banff, Alberta, Canada, 2004.

[5] J. Langford, L. Li, and T. Zhang. Sparse online learning via truncated gradient. *Journal of Machine Learning Research*, 10:777–801, 2009.

[6] Yu. Nesterov. Gradient methods for minimizing composiite objective function. CORE Discussion Paper 2007/76, Catholic University of Louvain, Center for Operations Research and Econometrics, 2007.

[7] P. Tseng. On accelerated proximal gradient methods for convex-concave optimization. Submitted to *SIAM Journal on Optimization*, 2008.

[8] A. Beck and M. Teboulle. A fast iterative shrinkage-threshold algorithm for linear inverse problems. Technical report, Technion, 2008. To appear in *SIAM Journal on Image Sciences*.

[9] M. Zinkevich. Online convex programming and generalized infinitesimal gradient ascent. In *Proceedings of the 20th International Conference on Machine Learning (ICML)*, pages 928–936, Washington DC, 2003.

[10] N. Cesa-Bianchi and G. Lugosi. *Predictioin, Learning, and Games*. Cambridge University Press, 2006.

[11] Yu. Nesterov. Primal-dual subgradient methods for convex problems. *Mathematical Programming*, 120(1):221–259, 2009. Appeared early as CORE discussion paper 2005/67, Catholic University of Louvain, Center for Operations Research and Econometrics.

[12] Yu. Nesterov. Smooth minimization of nonsmooth functions. *Mathematical Programming*, 103:127–152, 2005.

[13] L. Xiao. Dual averaging method for regularized stochastic learning and online optimization. Technical Report MSR-TR-2009-100, Microsoft Research, 2009.

[14] J. Duchi, S. Shalev-Shwartz, Y. Singer, and T. Chandra. Efficient projections onto the $\ell_1$-ball for learning in high dimensions. In *Proceedings of the 25th International Conference on Machine Learning (ICML)*, pages 272–279, 2008.

[15] P. Carbonetto, M. Schmidt, and N. De Freitas. An interior-point stochastic approximation method and an $l_1$-regularized delta rule. In D. Koller, D. Schuurmans, Y. Bengio, and L. Bottou, editors, *Advances in Neural Information Processing Systems 21*, pages 233–240. MIT Press, 2009.

[16] S. Balakrishnan and D. Madigan. Algorithms for sparse linear classifiers in the massive data setting. *Journal of Machine Learning Research*, 9:313–337, 2008.

[17] J. Duchi and Y. Singer. Efficient learning using forward-backward splitting. In *Proceedings of Neural Information Processing Systems*, December 2009.

[18] Y. LeCun, L. Bottou, Y. Bengio, and P. Haffner. Gradient-based learning applied to document recognition. *Proceedings of the IEEE*, 86(11):2278–2324, 1998. Dataset available at `http://yann.lecun.com/exdb/mnist`.

[19] K. Koh, S.-J. Kim, and S. Boyd. An interior-point method for large-scale $\ell_1$-regularized logistic regression. *Journal of Machine Learning Research*, 8:1519–1555, 2007.

[20] A. Nemirovski, A. Juditsky, G. Lan, and A. Shapiro. Robust stochastic approximation approach to stochastic programming. *SIAM Journal on Optimization*, 19(4):1574–1609, 2009.

